# Approximate Analytical Bootstrap Averages for Support Vector Classifiers

**Dörthe Malzahn**[1,2]          **Manfred Opper**[3]

[1] Informatics and Mathematical Modelling, Technical University of Denmark,
R.-Petersens-Plads, Building 321, Lyngby DK-2800, Denmark
[2] Institute of Mathematical Stochastics, University of Karlsruhe,
Englerstr. 2, Karlsruhe D-76131, Germany
[3] Neural Computing Research Group, School of Engineering and Applied Science,
Aston University, Birmingham B4 7ET, United Kingdom
malzahnd@isp.imm.dtu.dk          opperm@aston.ac.uk

## Abstract

We compute approximate analytical bootstrap averages for support vector classification using a combination of the replica method of statistical physics and the TAP approach for approximate inference. We test our method on a few datasets and compare it with exact averages obtained by extensive Monte-Carlo sampling.

## 1   Introduction

The bootstrap method [1, 2] is a widely applicable approach to assess the expected qualities of statistical estimators and predictors. Say, for example, in a supervised learning problem, we are interested in measuring the expected error of our favorite prediction method on test points [1] which are not contained in the training set $D_0$. If we have no hold out data, we can use the bootstrap approach to create artificial *bootstrap* data sets $D$ by resampling *with replacement* training data from the original set $D_0$. Each data point is taken with equal probability, i.e., some of the examples will appear several times in the bootstrap sample and others not at all. A proxy for the true average test error can be obtained by retraining the model on each bootstrap training set $D$, calculating the test error only on those points which are *not* contained in $D$ and finally averaging over all possible sets $D$.

While in general bootstrap averages can be approximated to any desired accuracy by the Monte-Carlo method, by generating a large enough number of random samples, it is useful to have also *analytical* approximations which avoid the time consuming retraining of the model for each new sample. Existing analytical approximations (based on asymptotic techniques) such as the *delta* method and the *saddle point* method require usually explicit analytical formulas for the estimators of the parameters for a trained model (see e.g. [3]). These may not be easily obtained for more complex models in machine learning such as support vector machines (SVMs). Recently, we introduced a novel approach for the approximate calculation of bootstrap averages [4] which avoids explicit formulas for parameter estimates. Instead, we define statistical estimators and predictors *implicitly*

as expectations with suitably defined pseudo-posterior Gibbs distributions over model parameters. Within this formulation, it becomes possible to perform averages over bootstrap samples *analytically* using the so-called "replica trick" of statistical physics [5]. The latter involves a specific analytic continuation of the original statistical model. After the average, we are left with a typically intractable inference problem for an effective Bayesian probabilistic model. As a final step, we use techniques for approximate inference to treat the probabilistic model. This combination of techniques allows us to obtain approximate bootstrap averages by solving a set of nonlinear equations rather than by explicit sampling.

Our method has passed a first test successfully on the simple case of Gaussian process (GP) regression, where explicit predictions are still cheaply computed. Also, since the original model is a smooth probabilistic one, the success of approximate inference techniques may be not too surprising. In this paper, we will address a more challenging problem, that of the *support vector machine*. In this case, the connection to a probabilistic model (a type of GP) can be only established by introducing a further parameter which must eventually *diverge* to obtain the SVM predictor. In this limit, the probabilistic model is becoming highly nonregular and approaches a deterministic model. Hence it is not clear *a priori* if our framework would survive these delicate limiting manipulations and still be able to give good approximate answers.

## 2  Hard Margin Support Vector Classifiers

The hard margin SVM is a classifier which predicts binary class labels $y = \text{sign}[\hat{f}_{D_0}(x)] \in \{-1, 1\}$ for inputs $x \in I\!R^d$ based on a set of training points $D_0 = (z_1, z_2, \ldots, z_N)$, where $z_i = (x_i, y_i)$ (for details see [6]). The usually nonlinear activation function $\hat{f}_{D_0}(x)$ (which we will call "internal field") is expressed as $\hat{f}_{D_0}(x) = \sum_{i=1}^{N} y_i \alpha_i K(x, x_i)$, where $K(x, x')$ is a positive definite kernel and the set of $\alpha_i$'s is computed from $D_0$ by solving a certain convex optimization problem.

For bootstrap problems, we fix the pool of training data $D_0$, and consider the statistics of vectors $\hat{\mathbf{f}}_D = (\hat{f}_D(x_1), \ldots, \hat{f}_D(x_N))$ at all inputs $x_i \in D_0$, when the predictor $\hat{f}$ is computed on randomly chosen subsets $D$ of $D_0$. Unfortunately, we *do not have an explicit analytical expression* for $\hat{\mathbf{f}}_D$, but it is obtained *implicitly* as the vector $\mathbf{f} = (f_1, \ldots, f_N)$ which solves the constraint optimization problem

$$\text{Minimize } \left\{ \mathbf{f}^T \mathbf{K}^{-1} \mathbf{f} \right\} \qquad \text{with } f_i y_i \geq 1 \qquad \text{for all } i \text{ such that } (x_i, y_i) \in D \quad (1)$$

$\mathbf{K}$ is the *kernel matrix* with elements $K(x_i, x_j)$.

## 3  Deriving Predictors from Gibbs Distributions

In this section, we will show how to obtain the SVM predictor $\hat{f}_D$ formally as the expectation over a certain type of Gibbs distribution over possible $\mathbf{f}$'s in the form

$$\hat{\mathbf{f}}_D = \langle \mathbf{f} \rangle = \int d\mathbf{f} \; \mathbf{f} \; P[\mathbf{f}|D] \tag{2}$$

with respect to a density $P[\mathbf{f}|D] = \frac{1}{Z}\mu[\mathbf{f}] \; P(D|\mathbf{f})$ which is constructed from a suitable prior distribution $\mu[\mathbf{f}]$, a certain type of "likelihood" $P(D|\mathbf{f})$ and a normalizing partition function

$$Z = \int d\mathbf{f} \; \mu[\mathbf{f}] \; P(D|\mathbf{f}) \,. \tag{3}$$

Our general notation suggests that this principle will apply to a variety of estimators and predictors of the MAP type.

To represent the SVM in this framework, we use a well established relation between SVM's and Gaussian process (GP) models (see e.g. [7, 8]). We choose the GP prior

$$\mu[\mathbf{f}] = \frac{1}{\sqrt{(2\pi)^N \beta^{-N} \det(\mathbf{K})}} \ \exp\left(-\frac{\beta}{2}\mathbf{f}^T \mathbf{K}^{-1} \mathbf{f}\right) \ . \tag{4}$$

The *pseudo-likelihood* [2] is defined by

$$P(D|\mathbf{f}) = \prod_{j:\, z_j \in D} P(z_j|f_j) = \prod_{j:\, z_j \in D} \Theta(y_j f_j - 1) \tag{5}$$

where $\Theta(u) = 1$ for $u > 0$ and $0$ otherwise. In the limit $\beta \to \infty$, the measure $P[\mathbf{f}|D] \propto \mu[\mathbf{f}]P(D|\mathbf{f})$ obviously concentrates at the vector $\hat{\mathbf{f}}$ which solves Eq. (1).

## 4  Analytical Bootstrap Averages Using the Replica Trick

With the bootstrap method, we would like to compute average properties of the estimator $\hat{\mathbf{f}}_D$, Eq. (2), when datasets $D$ are random subsamples of $D_0$. An important class of such averages are of the type of a *generalization error* $\varepsilon$ which are expectations of loss functions $g(\hat{f}_D(x_i); x_i, y_i)$ over *test* points $i$, i.e., those examples which are in $D_0$ but not contained in the bootstrap training set $D$. Hence, we define

$$\varepsilon \doteq \frac{1}{N}\sum_{i=1}^{N} \frac{E_D\left[\delta_{s_i,0}\, g(\hat{f}_D(x_i); x_i, y_i)\right]}{E_D\left[\delta_{s_i,0}\right]} \tag{6}$$

where $E_D[\cdots]$ denotes the expectation over random bootstrap samples $D$ which are created from the original training set $D_0$. Each sample $D$ is represented by a vector of "occupation" numbers $\mathbf{s} = (s_1, \ldots, s_N)$ where $s_i$ is the number of times example $z_i$ appears in the set $D$ and $\sum_{i=1}^{N} s_i = S$. The Kronecker symbol, defined by $\delta_{s_i,0} = 1$ for $s_i = 0$ and $0$ else, guarantees that only realizations of bootstrap training sets $D$ contribute to Eq. (6) which do not contain the test point. For fixed bootstrap sample size $S$, the distribution of $s_i$'s is multinomial. It is simpler (and does not make a big difference when $S$ is sufficiently large) when we work with a Poisson distribution for the size of the set $D$ with $S$ as the mean number of data points in the sample. Then we get the simpler, factorizing joint distribution

$$P(\mathbf{s}) = \prod_{i=1}^{N} \frac{(\frac{S}{N})^{s_i} e^{-S/N}}{s_i!} \tag{7}$$

for the occupation numbers $s_i$. From Eq. (7) we get $E_D[\delta_{s_i,0}] = e^{-\frac{S}{N}}$.

Since we can represent general loss functions $g$ by their Taylor expansions in powers of $\hat{f}_D$ (or polynomial approximations in case of non-smooth losses) it is sufficient to consider only monomials $g(\hat{f}_D(x); x, y) = (\hat{f}_D(x))^r$ for arbitrary $r$ in the following and regain the general case at the end by resumming the series. Using the definition of the estimator $\hat{\mathbf{f}}_D$, Eq. (2), the bootstrap expectation Eq. (6) can be rewritten as

$$\varepsilon(S) = \frac{1}{N}\sum_{i=1}^{N} \frac{E_D\left[\delta_{s_i,0}\, Z^{-r} \int \prod_{a=1}^{r}\left\{d\mathbf{f}^a\, \mu[\mathbf{f}^a]\, f_i^a\, \prod_{j=1}^{N}(P(z_j|f_j^a))^{s_j}\right\}\right]}{E_D\left[\delta_{s_i,0}\right]} \ . \tag{8}$$

which involves $r$ copies[3], i.e. *replicas* $\mathbf{f}^1, \ldots, \mathbf{f}^r$ of the parameter vector $\mathbf{f}$. If the partition functions $Z$ in the numerator of Eq. (8) were raised to *positive powers* rather than negative

ones, one could perform the bootstrap average over the distribution Eq. (7) analytically. To enable such an analytical average over the vector $\mathbf{s}$ (which is the "quenched disorder" in the language of statistical physics) one introduces the following "trick" extensively used in statistical physics of amorphous systems [5]. We introduce the auxiliary quantity

$$\varepsilon_n(S) = \frac{1}{e^{-\frac{S}{N}} N} \sum_{i=1}^{N} E_D \left[ \delta_{s_i,0} \; Z^{n-r} \int \prod_{a=1}^{r} \left\{ d\mathbf{f}^a \; \mu[\mathbf{f}^a] \; f_i^a \; \prod_{j=1}^{N} (P(z_j|f_j^a))^{s_j} \right\} \right] \quad (9)$$

for arbitrary real $n$, which allows to write

$$\varepsilon(S) = \lim_{n \to 0} \varepsilon_n(S). \quad (10)$$

The advantage of this definition is that for *integers* $n \geq r$, $\varepsilon_n(S)$ can be represented in terms of $n$ *replicas* $\mathbf{f}^1, \mathbf{f}^2, \ldots, \mathbf{f}^n$ of the original variable $\mathbf{f}$ for which an explicit average over $s_i$'s is possible. At the end of all calculations an analytical continuation to arbitrary real $n$ and the limit $n \to 0$ must be performed. For integer $n \geq r$, we use the definition of the partition function Eq. (3), exchange the expectation over datasets with the expectation over $\mathbf{f}$'s and use the explicit form of the distribution Eq. (7) to perform the average over bootstrap sets. The resulting expressions can be rewritten as [4]

$$\varepsilon_n(S) = \frac{\Xi_n^{\backslash i}}{N} \sum_{i=1}^{N} \left\langle\!\!\left\langle \prod_{a=1}^{r} f_i^a \right\rangle\!\!\right\rangle_{\backslash i}, \quad (11)$$

where $\langle\!\langle \cdots \rangle\!\rangle_{\backslash i}$ denotes an average with respect to the so called *cavity distribution* $P_{\backslash i}$ for replicated variables $\vec{f}_i = (f_i^1, \ldots, f_i^n)$ defined by

$$P_{\backslash i}(\vec{f}_i) \propto \frac{1}{L_i(\vec{f}_i)} \int \prod_{j=1, j \neq i}^{N} d\vec{f}_j \; P(\vec{f}_1, \ldots, \vec{f}_N) . \quad (12)$$

The joint distribution of replica variables $P(\vec{f}_1, \ldots, \vec{f}_N) \propto \prod_{a=1}^{n} \mu[\mathbf{f}^a] \; \prod_{j=1}^{N} L_j(\vec{f}_j)$ is defined by the new likelihoods

$$L_j(\vec{f}_j) = \exp\left[ -\frac{S}{N} \left( 1 - \prod_{a=1}^{n} P(z_j|f_j^a) \right) \right] . \quad (13)$$

## 5 TAP Approximation

We have mapped the original bootstrap problem to an inference problem for an effective Bayesian probabilistic model (the hidden variables have the dimensionality $N \times n$) for which we have to find a tractable approximation which allows analytical continuation of $n \to 0$ and $\beta \to \infty$. We use the adaptive TAP approach of Opper and Winther [9] which is often found to give more accurate results than, e.g., a simple mean field or a variational Gaussian approximation. The ADATAP approach replaces the analytically intractable cavity distribution Eq. (12) by a Gaussian distribution. In our case this can be written as

$$P_{\backslash i}(\vec{f}_i) \propto e^{-\frac{1}{2} \vec{f}^T \Lambda_c(i) \vec{f} + \gamma_c(i)^T \vec{f}} , \quad (14)$$

where the parameters $\Lambda_c$ and $\gamma_c$ are computed selfconsistently from the dataset $D_0$ by solving a set of coupled nonlinear equations. Details are given in the appendix.

The form Eq. (14) allows a simple way of dealing with the parameters $n$ and $\beta$. We utilize the exchangeability of variables $f_i^1, \ldots, f_i^n$ and assume replica symmetry and further

introduce an explicit scaling of all parameters with $\beta$. This scaling was found to make all final expressions finite in the limit $\beta \to \infty$. We set

$$
\begin{aligned}
\Lambda_c^{ab}(i) &= \Lambda_c(i) = \beta^2 \lambda_c(i) \quad \text{for } a \neq b \\
\Lambda_c^{aa}(i) &= \Lambda_c^0(i) = \beta^2 \lambda_c^0(i) \qquad \text{and} \qquad \gamma_c^a(i) = \beta \gamma_c(i) \ \text{ for all } \ a = 1, \dots, n \ .
\end{aligned}
\tag{15}
$$

We also assume that $\Delta \lambda_c(i) \doteq \beta^{-1}(\Lambda_c^0(i) - \Lambda_c(i))$ remains finite for $\beta \to \infty$. The ansatz Eq. (15) keeps the number of adjustable parameters independent of $n$ and allows to perform the "replica limit" $n \to 0$ and the "SVM-limit" $\beta \to \infty$ in all equations *analytically before* we start the final numerical parameter optimization.

Computing the expectation Eq. (11) with Eq. (14) and (15) and resumming the power series over $r$ yields the final theoretical expression for Eq. (6)

$$
\varepsilon(S) = \frac{1}{N} \sum_{i=1}^N \int dG(u) \, g\left( \frac{\gamma_c(i) + u\sqrt{-\lambda_c(i)}}{\Delta \lambda_c(i)}; x_i, y_i \right)
\tag{16}
$$

where $dG(u) = du(2\pi)^{-\frac{1}{2}} e^{-\frac{u^2}{2}}$ and $g$ is an arbitrary loss function. With $g(\hat{f}_D(x_i); x_i, y_i) = \Theta(-y_i \hat{f}_D(x_i))$ we obtain the bootstrapped classification error

$$
\varepsilon(S) = \frac{1}{N} \sum_{i=1}^N \Phi\left( -\frac{y_i \gamma_c(i)}{\sqrt{-\lambda_c(i)}} \right)
\tag{17}
$$

where $\Phi(x) = \int_{-\infty}^x dG(u)$.

Besides the computation of generalization errors, we can use our method to *quantify the uncertainty* of the SVM prediction at test points. This can be obtained by computing the bootstrap distribution of the "internal fields" $\hat{f}_D(x_i)$ at a *test* input $x_i$. This is obtained from Eq. (16) by inserting $g(\hat{f}_D(x_i); x_i, y_i) = \delta(\hat{f}_D(x_i) - h)$ using the Dirac $\delta$-function

$$
\rho_i(h) = \frac{\Delta \lambda_c(i)}{\sqrt{-2\pi \lambda_c(i)}} \exp\left( -\frac{(h \Delta \lambda_c(i) - \gamma_c(i))^2}{2(-\lambda_c(i))} \right) \ ,
\tag{18}
$$

i.e., $m_i^c = \frac{\gamma_c(i)}{\Delta \lambda_c(i)}$ and $V_{ii}^c = -\frac{\lambda_c(i)}{(\Delta \lambda_c(i))^2}$ are the predicted mean and variance of the internal field. (The predicted posterior variance of the internal field is $(\beta \Delta \lambda_c(i))^{-1}$ and goes to zero as $\beta \to \infty$ indicating the transition to a deterministic model.) It is possible to extend the result Eq. (18) to "real" test inputs $x \notin D_0$, which is of greater importance to applications. This replaces $\Delta \lambda_c(i)$, $\gamma_c(i)$, $\lambda_c(i)$ by

$$
\Delta \lambda_c(x) = \left( K(x,x) - \sum_{i=1}^N K(x, x_i) \Delta \lambda(i) T_i(x) \right)^{-1}
\tag{19}
$$

$$
\gamma_c(x) = \Delta \lambda_c(x) \sum_{i=1}^N T_i(x) \gamma(i)
$$

$$
\lambda_c(x) = (\Delta \lambda_c(x))^2 \sum_{i=1}^N (T_i(x))^2 \lambda(i)
$$

with $T_i(x) = \sum_{j=1}^N K(x, x_j)(\mathbf{I} + \text{diag}(\boldsymbol{\Delta\lambda})\mathbf{K})_{ji}^{-1}$. The parameters $\Delta\lambda(i)$, $\gamma(i)$, $\lambda(i)$ are determined from $D_0$ according to Eq. (22), (23).

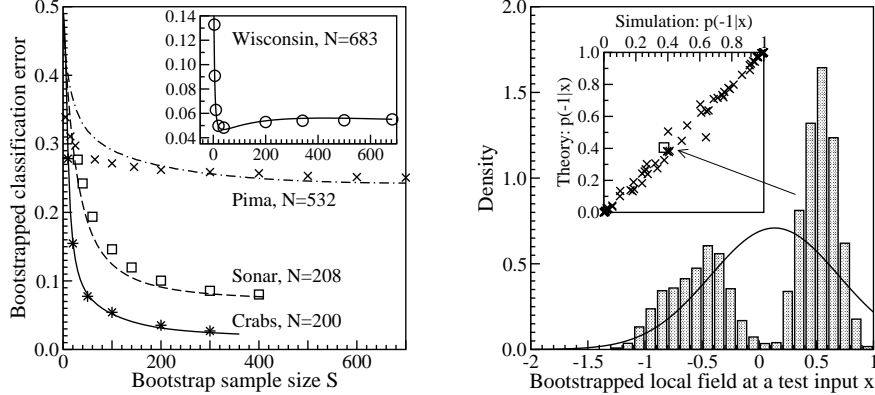

Figure 1: *Left:* Average bootstrapped generalization error for hard margin support vector classification on different data sets (simulation: symbols, theory: lines). *Right:* Boot-strapped distribution of the internal field for Sonar data at a *test* input $x \notin D_0$. Most distributions are Gaussian-like and in good agreement with the theory Eq. (18). We show an atypical case (simulation: histogram, theory line) which nevertheless predicts the relative weights for both class labels fairly well. The inset shows true versus estimated values of the probability $p(-1|x)$ for predicting label $y = -1$ .

## 6 Results for Bootstrap of Hard Margin Support Vector Classifiers

We determined the set of theoretical parameters by solving Eq. (21)-(23) for four bench-mark data sets $D_0$ [10] and different sample sizes $S$ using a RBF kernel $K(x, x') = \exp(-\frac{1}{2} \sum_{k=1}^{d} v_k (x_k - x'_k)^2))$ with individually customized hyperparameters $v_k$. The left panel of Fig.1 compares our theoretical results for the bootstrapped learning curves obtained by Eq. (17) (lines) with results from Monte-Carlo simulations (symbols). The Gaussian approximation of the cavity distribution is based on the assumption that the model prediction at a training input is influenced by a sufficiently large number of neighboring inputs. We expect it to work well for sufficiently broad kernel functions. This was the case for the Crabs and Wisconsin data sets where our theory is very accurate. It predicts correctly the interesting non-monotonous learning curve for the Wisconsin data (inset Fig.1, left). In comparison, the Sonar and Pima data sets were learnt with narrow RBF kernels. Here, we see that the quality of the TAP approximation becomes less good. However, our results provide still a reasonable estimate for the bootstrapped generalization error at sample size $S = N$. While for practical applications of estimating the "true" generalization error using *Efron's 0.632 bootstrap estimator* the case $S = N$ is of main importance, it is also interesting to discuss the limit of extreme oversampling $S \to \infty$. Since the hard margin SVM gains no additional information from the multiple presentation of the same data point, in this limit all bootstrap sets $D$ supply exactly the same information as the data set $D_0$ and the data average $E_D[\dots]$ becomes trivial. Variances with respect to $E_D[\dots]$ go to zero. With Eq. (21)-(23), we can write the average prediction $m_i$ at input $x_i \in D_0$ as $m_i = \sum_{j=1}^{N} y_j \alpha_j K(x_i, x_j)$ with weights $\alpha_j = \frac{\Delta\lambda(j)\Delta\lambda_c(j)}{\Delta\lambda(j)+\Delta\lambda_c(j)}(y_j m_j - y_j m_j^c)$ and recover for $S \to \infty$ the *Kuhn-Tucker conditions* $\alpha_i \geq 0$ and $\alpha_i \Theta(y_i m_i - 1) = 0$. The bootstrapped generalization error Eq. (17) is found to converge to the approximate leave-one-out error of Opper and Winther [8]

$$\lim_{S \to \infty} \varepsilon(S) = \frac{1}{N} \sum_{i=1}^{N} \Theta\left(-y_i m_i^c\right) = \sum_{i}^{SV} \Theta\left(\frac{\alpha_i}{[\mathbf{K}_{SV}^{-1}]_{ii}} - 1\right) \tag{20}$$

where the weights $\alpha_i$ are given by the SVM algorithm on $D_0$ and $\mathbf{K}_{SV}$ is the kernel matrix on the set of SV's. While the leave-one-out estimate is a *non-smooth* function of model parameters, Efron's $0.632\,\varepsilon(N)$ bootstrap estimate [2] of the generalization error approximated within our theory results in a *differentiable* expression Eq. (17) which may be used for kernel hyperparameter estimation. Preliminary results are promising.

The right panel of Fig. 1 shows results for the bootstrapped distribution of the internal field on *test* inputs $x \notin D_0$. The data set $D_0$ contained $N = 188$ Sonar data and the bootstrap is at sample size $S = N$. We find that the true distribution is often very Gaussian-like and well described by the theory Eq. (18). Figure 1 (right) shows a rare case where a bi-modal distribution (histogram) is found. Nevertheless, the Gaussian (line) predicted by our theory estimates the probability $p(-1|x)$ of a negative output quite accurately in comparison to the probability obtained from the simulation.

Both SVM training and the computation of our approximate SVM bootstrap requires the running of iterative algorithms. We compared the time $t^{train}$ for training a *single* SVM on each of the four benchmark data sets $D_0$ with the time $t^{theo}$ needed to solve our theory for SVM *bootstrap estimates* on these data for $S = N$. For sufficiently broad kernels we find $t^{train} \geq t^{theo}$ and our theory is reliable. The exception are extremely narrow kernels. For the latter (Pima example in Fig.1 (left)) we find $t^{theo} > t^{train}$ where our theory is still faster to compute but less reliable than a good Monte-Carlo estimate of the bootstrap.

## 7  Outlook

Our experiments on SVMs show that the approximate replica bootstrap approach appears to be highly robust to apply to models which only fit into our framework after some delicate limiting process. The SVM is also an important application because the prediction for each dataset requires the solution of a costly optimization problem. Experiments on benchmark data showed that our theory is appreciably faster to compute than a good Monte-Carlo estimate of the bootstrap and yields reliable results for kernels which are sufficiently broad. It will be interesting to apply our approach to other kernel methods such as *kernel PCA*. Since our method is based on a fairly general framework, we will also investigate if it can be applied to models where the bootstrapped parameters have a more complicated structure like, e.g., trees or hidden Markov models.

## Acknowledgments

DM gratefully acknowledges financial support by the Copenhagen Image and Signal Processing Graduate School and by the Postgraduate Programme "Natural Disasters" at the University of Karlsruhe.

## Appendix: TAP Equations

The ADATAP approach computes the set of parameters $\Lambda_c(i)$, $\gamma_c(i)$ by constructing an alternative set of tractable likelihoods $\hat{L}_j(\vec{f}) = e^{-\frac{1}{2}\vec{f}^T \Lambda(j)\vec{f} + \gamma(j)^T \vec{f}}$ defining an auxiliary Gaussian joint distribution $P_G(\vec{f}_1, \ldots, \vec{f}_N) \propto \prod_{a=1}^{n} \mu(\mathbf{f}^a) \prod_{j=1}^{N} \hat{L}_j(\vec{f}_j)$. We use replica symmetry and a specific scaling of the parameters with $\beta$: $\boldsymbol{\gamma}^a(j) = \beta\gamma(j)$, $\boldsymbol{\Lambda}^{aa}(j) = \Lambda^0(j) = \beta^2\lambda^0(j)$ for all $a$, $\boldsymbol{\Lambda}^{ab}(j) = \Lambda(j) = \beta^2\lambda(j)$ for $a \neq b$ and $\Delta\lambda(j) = \beta^{-1}(\Lambda^0(j) - \Lambda(j))$. All unknown parameters are found by *moment matching*: We assume that the first two marginal moments $m_i = \lim_{n \to 0} \langle\!\langle f_i^a \rangle\!\rangle$, $V_{ii} = \lim_{n \to 0} \langle\!\langle f_i^a f_i^b \rangle\!\rangle - (m_i)^2$, $\chi_{ii} = \beta \lim_{n \to 0} \langle\!\langle f_i^a f_i^a - f_i^a f_i^b \rangle\!\rangle$ of the variables $\vec{f}_i$ can be computed 1) by marginalizing $P_G$ and 2) by using the

relations between cavity distribution and marginal distributions $P(\vec{f_i}) \propto L_i(\vec{f_i}) P_{\backslash i}(\vec{f_i})$ as well as $P_G(\vec{f_i}) \propto \hat{L}_i(\vec{f_i}) P_{\backslash i}(\vec{f_i})$ for all $i = 1, \ldots, N$. This yields

$$\chi_{ii} = \chi_{ii}^c \left(1 - (1 - e^{-\frac{S}{N}})\Phi(\Delta_i^c)\right) \tag{21}$$

$$m_i = m_i^c \left(1 - (1 - e^{-\frac{S}{N}})\Phi(\Delta_i^c)\right) + y_i(1 - e^{-\frac{S}{N}})\left(\Phi(\Delta_i^c) + \frac{\sqrt{V_{ii}^c}}{\sqrt{2\pi}}e^{-\frac{1}{2}(\Delta_i^c)^2}\right)$$

$$V_{ii} = V_{ii}^c \left(1 - (1 - e^{-\frac{S}{N}})\Phi(\Delta_i^c)\right) + (1 - y_i m_i)(y_i m_i - y_i m_i^c)$$

where $m_i^c = \frac{\gamma_c(i)}{\Delta\lambda_c(i)}$, $V_{ii}^c = -\frac{\lambda_c(i)}{(\Delta\lambda_c(i))^2}$, $\chi_{ii}^c = \frac{1}{\Delta\lambda_c(i)}$ and $\Delta_i^c = \frac{1 - y_i m_i^c}{\sqrt{V_{ii}^c}}$. Further

$$\chi_{ii} = (\mathbf{G})_{ii} \tag{22}$$
$$m_i = (\mathbf{G}\,\boldsymbol{\gamma})_i$$
$$V_{ii} = -(\mathbf{G}\,\text{diag}(\boldsymbol{\lambda})\,\mathbf{G})_{ii}$$

with the $N \times N$ matrix $\mathbf{G} = (\mathbf{K}^{-1} + \text{diag}(\boldsymbol{\Delta\lambda}))^{-1}$ and

$$\chi_{ii} = \frac{1}{\Delta\lambda(i) + \Delta\lambda_c(i)} \tag{23}$$

$$m_i = \frac{\gamma(i) + \gamma_c(i)}{\Delta\lambda(i) + \Delta\lambda_c(i)}$$

$$V_{ii} = -\frac{\lambda(i) + \lambda_c(i)}{(\Delta\lambda(i) + \Delta\lambda_c(i))^2}$$

We solve Eq. (21)-(23) by iteration using Eqs. (21) and (22) to evaluate the moments $\{m_i, V_{ii}, \chi_{ii}\}$ and Eq. (23) to update the sets of parameters $\{\gamma_c(i), \Delta\lambda_c(i), \lambda_c(i)\}$ and $\{\gamma(i), \Delta\lambda(i), \lambda(i)\}$, respectively. Reasonable start values are $\Delta\lambda(i) = \Delta\lambda$, $\lambda(i) = -\Delta\lambda$, $\gamma(i) = y_i \Delta\lambda$ where $\Delta\lambda$ is obtained as the root of $0 = 1 - \frac{1}{N}\sum_{i=1}^N \frac{\omega_i \Delta\lambda}{1 + \omega_i \Delta\lambda} - (1 - (1 - e^{-S/N})\Phi(\Delta^c))$ with $\Delta^c = -0.5$ and $\omega_i$ are the eigenvalues of kernel matrix $\mathbf{K}$.

## Footnotes

[1]The average is over the unknown distribution of training data sets.

[2]It does not allow a full probabilistic interpretation [8].

[3]The superscripts should NOT be confused with powers of the variables.

[4] $P_{\backslash i}(\vec{f}_i)$, Eq. (12), has the normalizing partition function $\Xi_n^{\backslash i}$ where $\Xi_n^{\backslash i} \to 1$ for $n \to 0$.

## References

[1] B. Efron. *Ann. Statist.*, 7: 1-26, 1979.

[2] B. Efron, R. J. Tibshirani. *An Introduction to the Bootstrap*. Monographs on Statistics and Applied Probability 57, Chapman & Hall, 1993.

[3] J. Shao, D. Tu, *The Jackknife and Bootstrap*, Springer Series in Statistics, Springer, 1995.

[4] D. Malzahn, M. Opper, *A statistical mechanics approach to approximate analytical Bootstrap averages*, NIPS **15**, S. Becker, S. Thrun, K. Obermayer eds., MIT Press, 2003.

[5] M. Mézard, G. Parisi, M. A. Virasoro, *Spin Glass Theory and Beyond*, Lecture Notes in Physics **9**, World Scientific, 1987.

[6] B. Schölkopf, C. J. C. Burges, A. J. Smola (eds.), *Advances in Kernel Methods: Support Vector Learning*, MIT, Cambridge, MA, 1999.

[7] P. Sollich, *Probabilistic interpretation and Bayesian methods for Support Vector Machines*, In: ICANN99, pp.91-96, Springer 1999.

[8] M. Opper, O. Winther, *Neural Computation*, 12: 2655-2684, 2000.

[9] M. Opper, O. Winther, *Phys. Rev. Lett.*, 86: 3695, 2001.

[10] From `http://www1.ics.uci.edu/~mlearn/MLSummary.html` and `http://www.stats.ox.ac.uk/pub/PRNN/`.
